# TEMPORAL PATTERNS OF ACTIVITY IN NEURAL NETWORKS

Paolo Gaudiano
Dept. of Aerospace Engineering Sciences,
University of Colorado, Boulder CO 80309, USA

January 5, 1988

## Abstract

Patterns of activity over real neural structures are known to exhibit time–dependent behavior. It would seem that the brain may be capable of utilizing temporal behavior of activity in neural networks as a way of performing functions which cannot otherwise be easily implemented. These might include the origination of sequential behavior and the recognition of time–dependent stimuli. A model is presented here which uses neuronal populations with recurrent feedback connections in an attempt to observe and describe the resulting time-dependent behavior. Shortcomings and problems inherent to this model are discussed. Current models by other researchers are reviewed and their similarities and differences discussed.

## METHODS / PRELIMINARY RESULTS

In previous papers,[2,3] computer models were presented that simulate a net consisting of two spatially organized populations of realistic neurons. The populations are richly interconnected and are shown to exhibit internally sustained activity. It was shown that if the neurons have response times significantly shorter than the typical unit time characteristic of the input patterns (usually 1 msec), the populations will exhibit time–dependent behavior. This will typically result in the net falling into a limit cycle. By a limit cycle, it is meant that the population falls into activity patterns during which all of the active cells fire in a cyclic, periodic fashion. Although the period of firing of the individual cells may be different, after a fixed time the overall population activity will repeat in a cyclic, periodic fashion. For populations organized in 7x7 grids, the limit cycle will usually start 20-200 msec after the input is turned off, and its period will be in the order of 20-100 msec.

The point of interest is that if the net is allowed to undergo synaptic modifications by means of a modified hebbian learning rule while being presented with a specific spatial pattern (i.e., cells at specific spatial locations within the net are externally stimulated), subsequent presentations of the same pattern with different temporal characteristics will cause the population to recall patterns which are spatially identical (the same cells will be active) but which have different temporal qualities. In other words, the net can fall into a different limit cycle. These limit cycles seem to behave as attractors in that similar input patterns will result in the same limit cycle, and hence each distinct limit cycle appears to have a basin of attraction. Hence a net which can only learn a small

number of spatially distinct patterns can recall the patterns in a number of temporal modes. If it were possible to quantitatively discriminate between such temporal modes, it would seem reasonable to speculate that different limit cycles could correspond to different memory traces. This would significantly increase estimates on the capacity of memory storage in the net.

It has also been shown that a net being presented with a given pattern will fall and stay into a limit cycle until another pattern is presented which will cause the system to fall into a different basin of attraction. If no other patterns are presented, the net will remain in the same limit cycle indefinitely. Furthermore, the net will fall into the same limit cycle independently of the duration of the input stimulus, so long as the input stimulus is presented for a long enough time to raise the population activity level beyond a minimum necessary to achieve self-sustained activity. Hence, if we suppose that the net "recognizes" the input when it falls into the corresponding limit cycle, it follows that the net will recognize a string of input patterns regardless of the duration of each input pattern, so long as each input is presented long enough for the net to fall into the appropriate limit cycle. In particular, our system is capable of falling into a limit cycle within some tens of milliseconds. This can be fast enough to encode, for example, a string of phonemes as would typically be found in continuous speech. It may be possible, for instance, to create a model similar to Rumelhart and McClelland's 1981 model on word recognition by appropriately connecting multiple layers of these networks. If the response time of the cells were increased in higher layers, it may be possible to have the lowest level respond to stimuli quickly enough to distinguish phonemes (or some sub–phonemic basic linguistic unit), then have populations from this first level feed into a slower, word–recognizing population layer, and so on. Such a model may be able to perform word recognition from an input consisting of continuous phoneme strings even when the phonemes may vary in duration of presentation.

## SHORTCOMINGS

Unfortunately, it was noticed a short time ago that a consistent mistake had been made in the process of obtaining the above–mentioned results. Namely, in the process of decreasing the response time of the cells I accidentally reached a response time below the time step used in the numerical approximation that updates the state of each cell during a simulation. The equations that describe the state of each cell depend on the state of the cell at the previous time step as well as on the input at the present time. These equations are of first order in time, and an explicit discrete approximation is used in the model. Unfortunately it is a known fact that care must be taken in selecting the size of the time step in order to obtain reliable results. It is infact the case that by reducing the time step to a level below the response time of the cells the dynamics of the system varied significantly. It is questionable whether it would be possible to adjust some of the population parameters within reson to obtain the same results with a smaller step size, but the following points should be taken into account: 1) other researchers have created similar models that show such cyclic behavior (see for example Silverman, Shaw and Pearson[7]). 2) biological data exists which would indicate the existance of cyclic or periodic bahvior in real neural systems (see for instance Baird[1]).

As I just recently completed a series of studies at this university, I will not be able to perform a detailed examination of the system described here, but instead I will more

than likely create new models on different research equipment which will be geared more specifically towards the study of temporal behavior in neural networks.

## OTHER MODELS

It should be noted that in the past few years some researchers have begun investigating the possibility of neural networks that can exhibit time–dependent behavior, and I would like to report on some of the available results as they relate to the topic of temporal patterns. Baird[1] reports findings from the rabbit's olfctory bulb which indicate the existance of phase–locked oscillatory states corresponding to olfactory stimuli presented to the subjects. He outlines an elegant model which attributes pattern recognition abilities to competing instabilities in the dynamic activity of neural structures. He further speculates that inhomogeneous connectivity in the bulb can be selectively modified to achieve input–sensitive oscillatory states.

Silverman, Shaw and Pearson[7] have developed a model based on a biologically–inspired idealized neural structure, which they call the trion. This unit represents a localized group of neurons with a discrete firing period. It was found that small ensembles of trions with symmetric connections can exhibit quasi–stable periodic firing patterns which do not require pacemakers or external driving. Their results are inspired by existing physiological data and are consistent with other works.

Kleinfeld[6], and Sompolinsky and Kanter[8] independently developed neural network models that can generate and recognize sequential or cyclic patterns. Both models rely on what could be summarized as the recirculation of information through time–delayed channels.

Very similar results are presented by Jordan[4] who extends a typical connectionist or PDP model to include state and plan units with recurrent connections and feedback from output units through hidden units. He employs supervised learning with fuzzy constraints to induce learning of sequences in the system.

From a slightly different approach, Tank and Hopfield[9] make use of patterned sets of delays which effectively compress information in time. They develop a model which recognizes patterns by falling into local minima of a state–space energy function. They suggest that a systematic selection of delay functions can be done which will allow for time distortions that would be likely to occur in the input.

Finally, a somewhat different approach is taken by Homma, Atlas and Marks[5], who generalize a network for spatial pattern recognition to one that performs spatio–temporal patterns by extending classical principles from spatial networks to dynamic networks. In particular, they replace multiplication with convolution, weights with transfer functions, and thresholding with non linear transforms. Hebbian and Delta learning rules are similarly generalized. The resulting models are able to perform temporal pattern recognition.

The above is only a partial list of some of the relevant work in this field, and there are probably various other results I am not aware of.

## DISCUSSION

All of the above results indicate the importance of temporal patterns in neural networks. The need is apparent for further formal models which can successfully quantify temporal behavior in neural networks. Several questions must be answered to further

clarify the role and meaning of temporal patterns in neural nets. For instance, there is an apparent difference between a model that performs sequential tasks and one that performs recognition of dynamic patterns. It seems that appropriate selection of delay mechanisms will be necessary to account for many types of temporal pattern recognition. The question of scaling must also be explored: mechanism are known to exist in the brain which can cause delays ranging from the millisecond–range (e.g. variations in synaptic cleft size) to the tenth of a second range (e.g. axonal transmission times). On the other hand, the brain is capable of recognizing sequences of stimuli that can be much longer than the typical neural event, such as for instance being able to remember a song in its entirety. These and other questions could lead to interesting new aspects of brain function which are presently unclear.

# References

[1] Baird, B., "Nonlinear Dynamics of Pattern Formation and Pattern Recognition in the Rabbit Olfactory Bulb". Physica 22D, 150-175. 1986.

[2] Gaudiano, P., "Computer Models of Neural Networks". Unpublished Master's Thesis. University of Colorado. 1987.

[3] Gaudiano, P., MacGregor, R.J., "Dynamic Activity and Memory Traces in Computer-Simulated Recurrently-Connected Neural Networks". Proceedings of the First International Conference on Neural Networks. 2:177-185. 1987.

[4] Jordan, M.I., "Attractor Dynamics and Parallelism in a Connectionist Sequential Machine". Proceedings of the Eighth Annual Conference of the Cognitive Sciences Society. 1986.

[5] Homma, T., Atlas, L.E., Marks, R.J.II, "An Artificial Neural Network for Spatio-Temporal Bipolar Patterns: Application to Phoneme Classification". To appear in proceedings of Neural Information Processing Systems Conference (AIP). 1987.

[6] Kleinfeld, D., "Sequential State Generation by Model Neural Networks". Proc. Natl. Acad. Sci. USA. 83: 9469-9473. 1986.

[7] Silverman, D.J., Shaw, G.L., Pearson, J.C. "Associative Recall Properties of the Trion Model of Cortical Organization". Biol. Cybern. 53:259-271. 1986.

[8] Sompolinsky, H., Kanter, I. "Temporal Association in Asymmetric Neural Networks". Phys. Rev. Let. 57:2861-2864. 1986.

[9] Tank, D.W., Hopfield, J.J. "Neural Computation by Concentrating Information in Time". Proc. Natl. Acad. Sci. USA. 84:1896-1900. 1987.
